# Clustering Under Prior Knowledge with Application to Image Segmentation

**Mário A. T. Figueiredo**
*Instituto de Telecomunicações*
*Instituto Superior Técnico*
Technical University of Lisbon
**Portugal**

*mario.figueiredo@lx.it.pt*

**Dong Seon Cheng,    Vittorio Murino**
Vision, Image Processing, and Sound Laboratory
*Dipartimento di Informatica*
University of Verona
**Italy**

*cheng@sci.univr.it,  vittorio.murino@univr.it*

## Abstract

This paper proposes a new approach to model-based clustering under prior knowledge. The proposed formulation can be interpreted from two different angles: as penalized logistic regression, where the class labels are only indirectly observed (via the probability density of each class); as finite mixture learning under a grouping prior. To estimate the parameters of the proposed model, we derive a (generalized) EM algorithm with a closed-form E-step, in contrast with other recent approaches to semi-supervised probabilistic clustering which require Gibbs sampling or suboptimal shortcuts. We show that our approach is ideally suited for image segmentation: it avoids the combinatorial nature Markov random field priors, and opens the door to more sophisticated spatial priors (*e.g.*, wavelet-based) in a simple and computationally efficient way. Finally, we extend our formulation to work in unsupervised, semi-supervised, or discriminative modes.

## 1   Introduction

Most approaches to *semi-supervised learning* (SSL) see the problem from one of two (dual) perspectives: supervised classification with additional unlabelled data (see [20] for a recent survey); clustering with prior information or constraints (*e.g.*, [4, 10, 11, 15, 17]). The second perspective, usually termed *semi-supervised clustering* (SSC), is usually adopted when labels are totaly absent, but there are (usually pair-wise) relations that one wishes to enforce or encourage.

Most SSC techniques work by incorporating the constrains (or prior) into classical algorithms such as K-means or EM for mixtures. The *semi-supervision* may be hard (*i.e.*, grouping constraints [15, 17]), or have the form of a prior under which probabilistic clustering is performed [4, 11]. The later is clearly the most natural formulation for cases where one wishes to encourage, not enforce, certain relations; an obvious example is image segmentation, seen as clustering under a spatial prior, where neighboring sites should be encouraged, but not constrained, to belong to the same cluster/segment. However, the previous EM-type algorithms for this class of methods have a major drawback: the presence of the prior makes the E-step non-trivial, forcing the use of expensive Gibbs sampling [11] or suboptimal methods such as the *iterated conditional modes* algorithm [4].

In this paper, we introduce a new approach to mixture-based SSC, leading to a simple, fully deterministic, generalized EM (GEM) algorithm. The keystone is the formulation of SSC as a penalized logistic regression problem, where the labels are only indirectly observed. The linearity of the resulting complete log-likelihood, w.r.t. the missing group labels, underlies the simplicity of the resulting GEM algorithm. When applied to image segmentation, our method allows using spatial priors which are typical of image estimation problems (*e.g.*, restoration/denoising), such as Gaussian

fields or wavelet-based priors. Under these priors, the M-step of our GEM algorithm reduces to a simple image denoising procedure, for which there are several extremely efficient algorithms.

## 2  Formulation

We start from the standard formulation of finite mixture models: $\mathcal{X} = \{\mathbf{x}_1, ..., \mathbf{x}_n\}$ is an observed data set, where each $\mathbf{x}_i \in \mathrm{I\!R}^d$ was generated (independently) according to one of a set of $K$ probability (density or mass) functions $\{p(\cdot|\boldsymbol{\phi}^{(1)}), ..., p(\cdot|\boldsymbol{\phi}^{(K)})\}$. In image segmentation, each $\mathbf{x}_i$ is a pixel value (gray scale, $d = 1$; color, $d = 3$) or a vector of local (*e.g.*, texture) features. Associated with $\mathcal{X}$, there is a hidden label set $\mathcal{Y} = \{\mathbf{y}_1, ..., \mathbf{y}_n\}$, where $\mathbf{y}_i = [y_i^{(1)}, ..., y_i^{(K)}]^T \in \{0, 1\}^K$, with $y_i^{(k)} = 1$ if and only if $\mathbf{x}_i$ was generated by source $k$ (the so-called "1-of-K" binary encoding). Thus,

$$p(\mathcal{X}|\mathcal{Y}, \boldsymbol{\phi}) = \prod_{k=1}^{K} \prod_{i:\, y_i^{(k)}=1} p(\mathbf{x}_i|\boldsymbol{\phi}^{(k)}) = \prod_{i=1}^{n} \prod_{k=1}^{K} \left[ p(\mathbf{x}_i|\boldsymbol{\phi}^{(k)}) \right]^{y_i^{(k)}}, \tag{1}$$

where $\boldsymbol{\phi} = (\boldsymbol{\phi}^{(1)}, ..., \boldsymbol{\phi}^{(K)})$ is the set of parameters of the generative models of classes.

In standard mixture models, all the $\mathbf{y}_i$ are assumed to be independent and identically distributed samples following a multinomial distribution with probabilities $\{\eta^{(1)}, ..., \eta^{(K)}\}$, *i.e.*, $P(\mathcal{Y}) = \prod_i \prod_k (\eta^{(k)})^{y_i^{(k)}}$. This is the part of standard mixture models that has to be modified in order to insert grouping constraints [15] or a grouping prior $p(\mathcal{Y})$ [4, 11]. However, this prior destroys the simplicity of the standard E-step for finite mixtures, which is critically based on the independence assumption. We follow a different route to avoid that roadblock.

Let the hidden labels $\mathcal{Y} = \{\mathbf{y}_1, ..., \mathbf{y}_n\}$ depend on a new set of variables $\mathcal{Z} = \{\mathbf{z}_1, ..., \mathbf{z}_n\}$, where each $\mathbf{z}_i = [z_i^{(1)}, ..., z_i^{(K)}]^T \in \mathrm{I\!R}^K$ following a multinomial logistic model [5]:

$$P(\mathcal{Y}|\mathcal{Z}) = \prod_{i=1}^{n} \prod_{k=1}^{K} \left( P[y_i^{(k)} = 1|\mathbf{z}_i] \right)^{y_i^{(k)}}, \qquad \text{where} \qquad P[y_i^{(k)} = 1|\mathbf{z}_i] = \frac{e^{z_i^{(k)}}}{\sum_{l=1}^{K} e^{z_i^{(l)}}}. \tag{2}$$

Due to the normalization, we can set (w.l.o.g.) $z_i^{(K)} = 0$, for $i = 1, ..., n$ [5]. We're thus left with $n(K-1)$ real variables, *i.e.*, $\mathcal{Z} = \{\mathbf{z}^{(1)}, ..., \mathbf{z}^{(K-1)}\}$, where $\mathbf{z}^{(k)} = [z_1^{(k)}, ..., z_n^{(k)}]^T$; of course, $\mathcal{Z}$ can be seen as an $n \times (K-1)$ matrix, where $\mathbf{z}^{(k)}$ is the $k$-th column and $\mathbf{z}_i$ is the $i$-th row.

With this formulation, certain grouping preferences may be expressed by a prior $p(\mathcal{Z})$. For example, preferred pair-wise relations can be easily embodied in a Gaussian prior

$$p(\mathcal{Z}) \propto \prod_{k=1}^{K-1} \exp\left[ -\frac{1}{4} \sum_{i=1}^{n} \sum_{j=1}^{n} A_{i,j}(z_i^{(k)} - z_j^{(k)})^2 \right] = \prod_{k=1}^{K-1} \exp\left[ -\frac{1}{2} (\mathbf{z}^{(k)})^T \boldsymbol{\Delta}\, \mathbf{z}^{(k)} \right], \tag{3}$$

where $\mathbf{A}$ is a matrix (with a null diagonal) encoding pair-wise preferences ($A_{i,j} > 0$ expresses preference, with strength proportional to $A_{i,j}$, for having points $i$ and $j$ in the same cluster) and $\boldsymbol{\Delta}$ is the well-known graph-Laplacian matrix [20],

$$\boldsymbol{\Delta} = \text{diag}\left\{ \textstyle\sum_{j=1}^{n} A_{1,j}, ..., \sum_{j=1}^{n} A_{n,j} \right\} - \mathbf{A}. \tag{4}$$

For image segmentation, each $\mathbf{z}^{(k)}$ is an image with real-valued elements and a natural choice for $\mathbf{A}$ is to have $A_{i,j} = \lambda$, if $i$ and $j$ are neighbors, and zero otherwise. Assuming periodic boundary conditions for the neighborhood system, $\boldsymbol{\Delta}$ is a block-circulant matrix with circulant blocks [2]. However, as shown below, other more sophisticated priors (such as wavelet-based priors) can also be used at no additional computational cost [1].

## 3  Model Estimation

### 3.1  Marginal Maximum A Posteriori and the GEM Algorithm

Based on the formulation presented in the previous section, SSC is performed by estimating $\mathcal{Z}$ and $\boldsymbol{\phi}$, seeing $\mathcal{Y}$ as missing data. The marginal *maximum a posteriori* estimate is obtained by

marginalizing out the hidden labels (over all the possible label configurations),

$$\left(\widehat{\mathcal{Z}}, \widehat{\phi}\right) = \arg\max_{\mathcal{Z}, \phi} \sum_{\mathcal{Y}} p(\mathcal{X}, \mathcal{Y}, \mathcal{Z}|\phi) = \arg\max_{\mathcal{Z}, \phi} \sum_{\mathcal{Y}} p(\mathcal{X}|\mathcal{Y}, \phi)\, P(\mathcal{Y}|\mathcal{Z})\, p(\mathcal{Z}), \qquad (5)$$

where we're assuming a flat prior for $\phi$. One of the key advantages of this approach is that (5) is a continuous (not combinatorial) optimization problem. This is in contrast Markov random field approaches to image segmentation, which lead to hard combinatorial problems, since they perform optimization directly with respect to the (discrete) label variables $\mathcal{Y}$. Finally, notice that once in possession of an estimate $\widehat{\mathcal{Z}}$, one may compute $P(\mathcal{Y}|\widehat{\mathcal{Z}})$ which gives the probability that each data point belongs to each class. By finding $\arg\max_k P[y_i^{(k)} = 1|\mathbf{z}_i]$, for every $i$, one may obtain a hard clustering/segmentation.

We handle (5) with a generalized EM (GEM) algorithm [13], *i.e.*, by applying the following iterative procedure (until some convergence criterion is satisfied):

**E-step:** Compute the conditional expectation of the complete log-posterior, given the current estimates $(\widehat{\mathcal{Z}}, \widehat{\phi})$ and the observations $\mathcal{X}$:

$$Q(\mathcal{Z}, \phi|\widehat{\mathcal{Z}}, \widehat{\phi}) = E_{\mathcal{Y}}\left[\log p(\mathcal{Y}, \mathcal{Z}, \phi|\mathcal{X})\,\Big|\,\widehat{\mathcal{Z}}, \widehat{\phi}, \mathcal{X}\right]. \qquad (6)$$

**M-step:** Update the estimate: $(\widehat{\mathcal{Z}}, \widehat{\phi}) \leftarrow (\widehat{\mathcal{Z}}_{\text{new}}, \widehat{\phi}_{\text{new}})$, with new values such that

$$Q(\widehat{\mathcal{Z}}_{\text{new}}, \widehat{\phi}_{\text{new}}|\widehat{\mathcal{Z}}, \widehat{\phi}) \geq Q(\widehat{\mathcal{Z}}, \widehat{\phi}|\widehat{\mathcal{Z}}, \widehat{\phi}). \qquad (7)$$

Under mild conditions, it is well known that GEM algorithms converge to a local maximum of the marginal log-posterior [18].

### 3.2 E-step

The complete log-posterior is

$$\log p(\mathcal{Y}, \mathcal{Z}, \phi|\mathcal{X}) \doteq \log p(\mathcal{X}|\mathcal{Y}, \phi) + \log P(\mathcal{Y}|\mathcal{Z}) + \log p(\mathcal{Z})$$
$$\doteq \sum_{i=1}^{n}\sum_{k=1}^{K} y_i^{(k)} \log p(\mathbf{x}_i|\phi^{(k)}) + \sum_{i=1}^{n}\left[\sum_{k=1}^{K} y_i^{(k)} z_i^{(k)} - \log\sum_{k=1}^{K} e^{z_i^{(k)}}\right] + \log p(\mathcal{Z}) \qquad (8)$$

where $\doteq$ stands for "equal up to an additive constant". The key observation is that this function is linear w.r.t. the hidden variables $y_i^{(k)}$. Consequently, the E-step reduces to computing their conditional expectations, which are then plugged into (8).

As in standard mixtures, each missing $y_i^{(k)}$ is binary, thus its expectation (denoted $\widehat{y}_i^{(k)}$) equals its posterior probability of being equal to one, easily obtained via Bayes law:

$$\widehat{y}_i^{(k)} \equiv E[y_i^{(k)}|\widehat{\mathcal{Z}}, \widehat{\phi}, \mathcal{X}] = P[y_i^{(k)} = 1|\widehat{\mathbf{z}}_i, \widehat{\phi}, \mathbf{x}_i] = \frac{p(\mathbf{x}_i|\widehat{\phi}^{(k)})\, P[y_i^{(k)} = 1|\widehat{\mathbf{z}}_i]}{\sum_{j=1}^{K} p(\mathbf{x}_i|\widehat{\phi}^{(j)})\, P[y_i^{(j)} = 1|\widehat{\mathbf{z}}_i]}. \qquad (9)$$

Notice that this is the same as the E-step for a standard finite mixture, where the probabilities $P[y_i^{(k)} = 1|\widehat{\mathbf{z}}_i]$ (given by (2)) play the role of the probabilities of the classes/components. Finally, the $Q$ function is obtained by plugging the expectations $\widehat{y}_i^{(k)}$ into (8).

### 3.3 M-Step

It's clear from (8) that the maximization w.r.t. $\phi$ can be performed separately w.r.t. to each $\phi^{(k)}$,

$$\widehat{\phi}_{\text{new}}^{(k)} = \arg\max_{\phi^{(k)}} \sum_{i=1}^{n} \widehat{y}_i^{(k)} \log p(\mathbf{x}_i|\phi^{(k)}). \qquad (10)$$

This is the familiar weighted maximum likelihood criterion, exactly as it appears in EM for standard mixtures. The explicit form of this update depends on the choice of $p(\cdot|\phi^{(k)})$; *e.g.*, this step can be easily applied to any finite mixture of exponential family densities [3].

In supervised image segmentation, these parameters are known (*e.g.*, previously estimated from training data) and thus it's not necessary to estimate them; the M-step reduces to the estimation of $\mathcal{Z}$. In unsupervised image segmentation, $\phi$ is unknown and (10) will have to be applied.

To update the estimate of $\mathcal{Z}$, we need to maximize (or at least improve, see (7))

$$L(\mathcal{Z}|\widehat{\mathcal{Y}}) \equiv \sum_{i=1}^{n} \left[ \sum_{k=1}^{K} \widehat{y}_i^{(k)} z_i^{(k)} - \log \sum_{k=1}^{K} e^{z_i^{(k)}} \right] + \log p(\mathcal{Z}). \tag{11}$$

Without the prior, this would be a simple *logistic regression* (LR) problem, with an identity design matrix [5]; however, instead of the usual hard labels $y_i^{(k)} \in \{0, 1\}$, we have "soft" labels $\widehat{y}_i^{(k)} \in [0, 1]$.

Arguably, the two standard approaches to maximum likelihood LR are the Newton-Raphson algorithm (a.k.a. *iteratively reweighted least squares* – IRLS [7]) and the *minorize-maximize* (MM) approach (formerly known as *bound optimization*) [5, 9]. We will show below that the MM approach can be easily modified to accommodate the presence of a prior.

Let's briefly review the MM approach for maximizing a twice differentiable concave function $E(\boldsymbol{\theta})$ with bounded Hessian [5, 9]. Let the Hessian $\mathcal{H}(\boldsymbol{\theta})$ of $E(\boldsymbol{\theta})$ be bounded below by $-\mathbf{B}$ (that is, $\mathcal{H}(\boldsymbol{\theta}) \succeq -\mathbf{B}$, in the matrix sense, meaning that $\mathcal{H}(\boldsymbol{\theta}) + \mathbf{B}$ is positive definite), where $\mathbf{B}$ is a positive definite matrix. It's trivial to show that $E(\boldsymbol{\theta}) - R(\boldsymbol{\theta}, \widehat{\boldsymbol{\theta}})$ has a minimum at $\boldsymbol{\theta} = \widehat{\boldsymbol{\theta}}$, where

$$R(\boldsymbol{\theta}, \widehat{\boldsymbol{\theta}}) = -\frac{1}{2} \left( \boldsymbol{\theta} - \widehat{\boldsymbol{\theta}} - \mathbf{B}^{-1}\mathbf{g}(\widehat{\boldsymbol{\theta}}) \right)^{T} \mathbf{B} \left( \boldsymbol{\theta} - \widehat{\boldsymbol{\theta}} - \mathbf{B}^{-1}\mathbf{g}(\widehat{\boldsymbol{\theta}}) \right), \tag{12}$$

with $\mathbf{g}(\widehat{\boldsymbol{\theta}})$ denoting the gradient of $E(\boldsymbol{\theta})$ at $\widehat{\boldsymbol{\theta}}$. Thus, the iteration

$$\widehat{\boldsymbol{\theta}}_{\text{new}} = \arg \max_{\boldsymbol{\theta}} R(\boldsymbol{\theta}, \widehat{\boldsymbol{\theta}}) = \widehat{\boldsymbol{\theta}} + \mathbf{B}^{-1}\mathbf{g}(\widehat{\boldsymbol{\theta}}) \tag{13}$$

is guaranteed to monotonically improve $E(\boldsymbol{\theta})$, *i.e.*, $E(\widehat{\boldsymbol{\theta}}_{\text{new}}) \geq E(\widehat{\boldsymbol{\theta}})$.

It was shown in [5] that the gradient and the Hessian of the logistic log-likelihood function, *i.e.*, (11) without the log-prior, verify (with $\mathbf{I}_a$ denoting an $a \times a$ identity matrix and $\mathbf{1}_a$ a vector of $a$ ones)

$$\mathbf{g}(\mathbf{z}) = \widehat{\mathbf{y}} - \boldsymbol{\eta}(\mathbf{z}) \quad \text{and} \quad \mathcal{H}(\mathbf{z}) \succeq -\frac{1}{2} \left( \mathbf{I}_{K-1} - \frac{\mathbf{1}_{K-1}\, \mathbf{1}_{K-1}^{T}}{K} \right) \otimes \mathbf{I}_n \equiv -\mathbf{B}, \tag{14}$$

where $\mathbf{z} = [z_1^{(1)}, ..., z_n^{(1)}, z_1^{(2)}, ..., z_n^{(K-1)}]^T$ denotes the lexicographic vectorization of $\mathcal{Z}$, $\widehat{\mathbf{y}}$ denotes the corresponding lexicographic vectorization of $\widehat{\mathcal{Y}}$, and $\boldsymbol{\eta}(\mathbf{z}) = [p_1^{(1)}, ..., p_n^{(1)}, p_1^{(2)}, ..., p_n^{(K-1)}]^T$ with $p_i^{(k)} = P[y_i^{(k)} = 1 | \mathbf{z}_i]$.

Defining $\mathbf{v} = \widehat{\mathbf{z}} + \mathbf{B}^{-1}(\widehat{\mathbf{y}} - \boldsymbol{\eta}(\widehat{\mathbf{z}}))$, the MM update equation for solving (11) is thus

$$\widehat{\mathbf{z}}_{\text{new}}(\mathbf{v}) = \arg \min_{\mathbf{z}} \left\{ \frac{1}{2} (\mathbf{z} - \mathbf{v})^T \mathbf{B} (\mathbf{z} - \mathbf{v}) - \log p(\mathbf{z}) \right\}, \tag{15}$$

where $p(\mathbf{z})$ is equivalent to $p(\mathcal{Z})$, because $\mathbf{z}$ is simply the lexicographic vectorization of $\mathcal{Z}$.

We now summarize our GEM algorithm:

**E-step:** compute $\widehat{\mathbf{y}}$, using (9), for all $i = 1, ..., n$ and $k = 1, ..., K - 1$.

**(Generalized) M-step:** Apply one or more iterations (15), keeping $\widehat{\mathbf{y}}$ fixed, that is, loop through the following two steps: $\mathbf{v} \leftarrow \widehat{\mathbf{z}} + \mathbf{B}^{-1}(\widehat{\mathbf{y}} - \boldsymbol{\eta}(\widehat{\mathbf{z}}))$ and $\widehat{\mathbf{z}} \leftarrow \widehat{\mathbf{z}}_{\text{new}}(\mathbf{v})$.

### 3.4 Speeding Up the Algorithm

In image segmentation, the MM update equation (15) is formally equivalent to the MAP estimation of an image with $n$ pixels in $I\!R^{K-1}$, under prior $p(\mathbf{z})$, where $\mathbf{v}$ plays the role of observed image, and $\mathbf{B}$ is the inverse covariance matrix of the noise. Due to the structure of $\mathbf{B}$, even if the prior models the several $\mathbf{z}^{(k)}$ as independent, *i.e.*, if $\log p(\mathbf{z}) = \log p(\mathbf{z}^{(1)}) + \cdots + \log p(\mathbf{z}^{(K-1)})$, (15) can not be decoupled into the several components $\{\mathbf{z}^{(1)}, ..., \mathbf{z}^{(K-1)}\}$. We sidestep this difficulty, at the cost of using a less tight bound in (14), based the following lemma:

**Lemma 1** *Let $\xi_K = 1/2$, if $K > 2$, and $\xi_K = 1/4$, if $K = 2$. Then, $\mathbf{B} \preceq \xi_K \, \mathbf{I}_{n(K-1)}$.*

**Proof:** Inserting $K = 2$ in (14) yields $\mathbf{B} = \mathbf{I}/4$, which proves the case $K = 2$. For $K > 2$, the inequality $\mathbf{I}/2 \succeq \mathbf{B}$ is equivalent to $\lambda_{\min}(\mathbf{I}/2 - \mathbf{B}) \geq 0$, which is equivalent to $\lambda_{\max}(\mathbf{B}) \leq (1/2)$. Since the eigenvalues of the Kronecker product are the products of the eigenvalues of the matrices, $\lambda_{\max}(\mathbf{B}) = \lambda_{\max}(\mathbf{I} - (1/K)\,\mathbf{1}\,\mathbf{1}^T)/2$. Since $\mathbf{1}\,\mathbf{1}^T$ is a rank-1 matrix with eigenvalues $\{0, ..., 0, K-1\}$, the eigenvalues of $(\mathbf{I} - (1/K)\,\mathbf{1}\,\mathbf{1}^T)$ are $\{1, ..., 1, 1/K\}$, thus $\lambda_{\max}(\mathbf{I} - (1/K)\,\mathbf{1}\,\mathbf{1}^T) = 1$, and $\lambda_{\max}(\mathbf{B}) = 1/2$. ∎

This lemma allows replacing $\mathbf{B}$ with $\xi_K \, \mathbf{I}_{n(K-1)}$ in (15) which (assuming independent priors, as is the case of (3)) becomes decoupled, leading to

$$\widehat{\mathbf{z}}_{\text{new}}^{(k)}(\mathbf{v}^{(k)}) = \arg\min_{\mathbf{z}^{(k)}} \left\{ \frac{\xi_K}{2} \left\| \mathbf{z}^{(k)} - \mathbf{v}^{(k)} \right\|^2 - \log p(\mathbf{z}^{(k)}) \right\}, \quad \text{for} \quad k = 1, ..., K-1, \quad (16)$$

where $\mathbf{v}^{(k)} = \widehat{\mathbf{z}}^{(k)} + (1/\xi_K)(\widehat{\mathbf{y}}^{(k)} - \boldsymbol{\eta}^{(k)}(\widehat{\mathbf{z}}^{(k)}))$. Moreover, the "noise" in each of these "denoising" problems is white and Gaussian, of variance $1/\xi_K$.

## 3.5 Stationary Gaussian Field Priors

Consider a Gaussian prior of form (3), where $A_{i,j}$ only depends on the relative position of $i$ and $j$ and the neighborhood system defined by $\mathbf{A}$ has periodic boundary conditions. In this case, both $\mathbf{A}$ and $\boldsymbol{\Delta}$ are block-circulant matrices, with circulant blocks [2], thus diagonalizable by a 2D discrete Fourier transform (2D-DFT). Formally, $\boldsymbol{\Delta} = \mathbf{U}^H \mathbf{D} \mathbf{U}$, where $\mathbf{D}$ is diagonal, $\mathbf{U}$ is the orthogonal matrix representing the 2D-DFT, and $(\cdot)^H$ denotes conjugate transpose. The log-prior is then expressed in the DFT domain, $\log p(\mathbf{z}^{(k)}) \doteq \frac{1}{2}(\mathbf{U}\mathbf{z}^{(k)})^H \mathbf{D}(\mathbf{U}\mathbf{z}^{(k)})$, and the solution of (16) is

$$\widehat{\mathbf{z}}_{\text{new}}^{(k)}(\mathbf{v}^{(k)}) = \xi_K \, \mathbf{U}^H \left[ \xi_K \mathbf{I}_n + \mathbf{D} \right]^{-1} \mathbf{U} \, \mathbf{v}^{(k)}, \quad \text{for} \quad k = 1, ..., K-1. \quad (17)$$

Observe that (17) corresponds to filtering each image $\mathbf{v}^{(k)}$, in the DFT domain, with a fixed filter with frequency response $[\xi_K \mathbf{I}_n + \mathbf{D}]^{-1}$; this inversion can be computed off-line and is trivial because $\xi_K \mathbf{I}_n + \mathbf{D}$ is diagonal. Finally, it's worth stressing that the matrix-vector products by $\mathbf{U}$ and $\mathbf{U}^H$ are not carried out explicitly but more efficiently via the FFT algorithm, with cost $O(n \log n)$.

## 3.6 Wavelet-Based Priors for Segmentation

It's known that piece-wise smooth images have sparse wavelet-based representations (see [12] and the many references therein); this fact underlies the state-of-the-art denoising performance of wavelet-based methods. Piece-wise smoothness of the $\mathbf{z}^{(k)}$ translates into segmentations in which pixels in each class tend to form connected regions. Consider a wavelet expansion of each $\mathbf{z}^{(k)}$

$$\mathbf{z}^{(k)} = \mathbf{W}\boldsymbol{\theta}^{(k)}, \quad k = 1, ..., K-1, \quad (18)$$

where the $\boldsymbol{\theta}^{(k)}$ are sets of coefficients and $\mathbf{W}$ is the matrix representation of an inverse wavelet transform; $\mathbf{W}$ may be orthogonal or have more columns than lines (over-complete representations) [12]. A wavelet-based prior for $\mathbf{z}^{(k)}$ is induced by placing a prior on the coefficients $\boldsymbol{\theta}^{(k)}$. A classical choice for $p(\boldsymbol{\theta}^{(k)})$ is a generalized Gaussian [14]. Without going into details, under this class of priors (and others), (16) becomes a non-linear wavelet-based denoising step, which has been widely studied in the image processing literature. For several choices of $p(\boldsymbol{\theta}^{(k)})$ and $\mathbf{W}$, this denoising step has a very simple closed form, which essentially corresponds to computing a wavelet transform of the observations, applying a coefficient-wise non-linear shrinkage/thresholding operation, and applying the inverse transform to the processed coefficients. This is computationally very efficient, due to the existence of fast algorithms for computing direct and inverse wavelet transforms; *e.g.*, $O(n)$ for an orthogonal wavelet transform or $O(n \log n)$ for a shift-invariant redundant transform.

# 4 Extensions

## 4.1 Semi-Supervised Segmentation

For semi-supervised image segmentation, the user defines regions in the image for which the true label is known. Our GEM algorithm is trivially modified to handle this case: if at location $i$ the label

is known to be (say) $k$, we freeze $\widehat{y}_i^{(k)} = 1$, and $\widehat{y}_i^{(j)} = 0$, for $j \neq k$. The E-step is only applied to those locations for which the label is unknown. The M-step remains unchanged.

## 4.2   Discriminative Features

Our formulation (as most probabilistic segmentation methods) adopts a generative perspective, where each $p(\cdot|\phi^{(k)})$ models the data generation mechanism in the corresponding class. However, discriminative methods (such as support vector machines) are seen as the current state-of-the-art in classification [7]. We will now show how a pre-trained discriminative classifier can be used in our GEM algorithm instead of the generative likelihoods.

The E-step (see (9)) obtains the posterior probability that $\mathbf{x}_i$ was generated by the $k$-th model, by combining (via Bayes law) the corresponding likelihood $p(\mathbf{x}_i|\widehat{\phi}^{(k)})$ with the local prior probability $P[y_i^{(k)} = 1|\widehat{\mathbf{z}}_i]$. Consider that, instead of likelihoods derived from generative models, we have a discriminative classifier, $i.e.$, one that directly provides estimates of the posterior class probabilities $P[y_i^{(k)} = 1|\mathbf{x}_i]$. To use these values in our segmentation algorithm, we need a way to bias these estimates according to the local prior probabilities $P[y_i^{(k)} = 1|\widehat{\mathbf{z}}_i]$, which are responsible for encouraging spatial coherence. Let us assume that we know that the discriminative classifier was trained using $m_k$ samples from the $k$-th class. It can thus be assumed that these posterior class probabilities verify $P[y_i^{(k)} = 1|\mathbf{x}_i] \propto m_k \, p(\mathbf{x}_i|y_i^{(k)} = 1)$. It is then possible to "bias" these classifiers, with the local prior probabilities $P[y_i^{(k)} = 1|\widehat{\mathbf{z}}_i]$, simply by computing

$$P[y_i^{(k)} = 1|\mathbf{x}_i] = \frac{P[y_i^{(k)} = 1|\mathbf{x}_i] \, P[y_i^{(k)} = 1|\widehat{\mathbf{z}}_i]}{m_k} \left( \sum_{j=1}^{K} \frac{P[y_i^{(j)} = 1|\mathbf{x}_i] \, P[y_i^{(j)} = 1|\widehat{\mathbf{z}}_i]}{m_j} \right)^{-1}.$$

# 5   Experiments

In this section we will show experimental results of image segmentation in supervised, unsupervised, semi-supervised, and discriminative modes. Assessing the performance of a segmentation method is not a trivial task. Moreover, the performance of segmentation algorithms depends more critically on the adopted features (which is not the focus of this paper) than on the spatial coherence prior. For these reasons, we will not present any careful comparative study, but simply a set of experimental examples testifying for the promising behavior of the proposed approach.

## 5.1   Supervised and Unsupervised Image Segmentation

The first experiment, reported in Fig. 1, illustrates the algorithm on a synthetic gray scale image with four Gaussian classes of means 1, 2, 3, and 4, and standard deviation 0.6. For this image, both supervised and unsupervised segmentation lead to almost visually indistinguishable results, so we only show the supervised segmentation results. In the Gaussian prior, matrix $\mathbf{A}$ corresponds to a first order neighborhood, that is, $A_{i,j} = \gamma$ if and only if $j$ is one of the four nearest neighbors of $i$. For wavelet-based segmentation, we have used undecimated Haar wavelets and the Bayes-shrink denoising procedure [6].

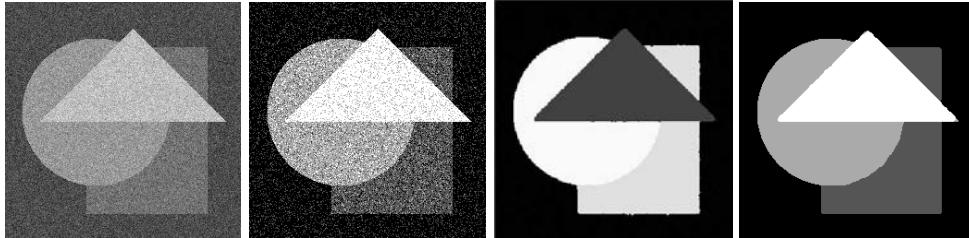

Figure 1: From left to right: observed image, maximum likelihood segmentation, GEM result with Gaussian prior, GEM result with wavelet-based prior.

## 5.2 Semi-supervised Image Segmentation

We illustrate the semi-supervised mode of our approach on two real RGB images, shown in Fig. 2. Each region is modelled by a single multivariate Gaussian density in RGB space. In the example in the first row, the goal is to segment the image into skin, cloth, and background regions; in the second example, the goal is to segment the horses from the background. These examples show how the semi-supervised mode of our algorithm is able to segment the image into regions which "look like" the seed regions provided by the user.

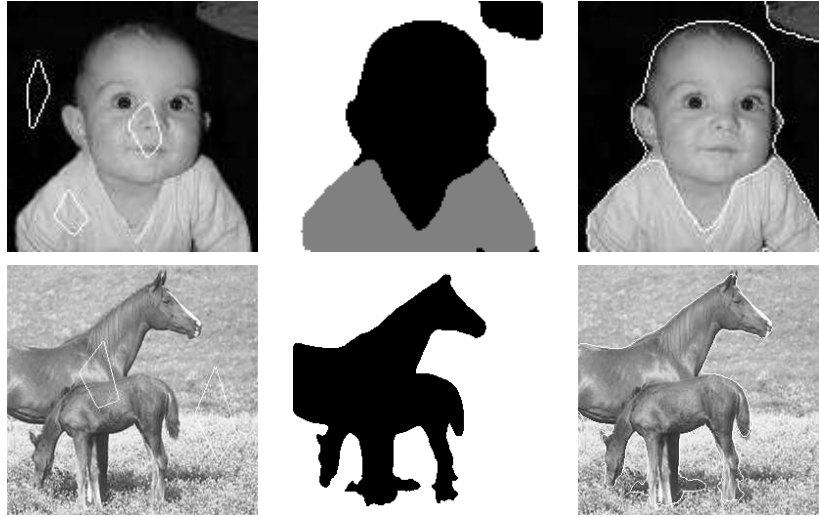

Figure 2: From left to right (in each row): observed image with regions indicated by the user as belonging to each class, segmentation result, region boundaries.

## 5.3 Discriminative Texture Segmentation

Finally, we illustrate the behavior of the algorithm when used with discriminative classifiers by applying it to texture segmentation. We build on the work in [8], where SVM classifiers are used for texture classification (see [8] for complete details about the kernels and texture features used). Fig. 3 shows two experiments; one with a two-texture $256 \times 512$ image and the other with a 5-texture $256 \times 256$ image. In the two-class case, one binary SVM was trained on $1000$ random patterns from each class. For the 5-class case, 5 binary SVMs were trained in the "1-vs-all" mode, with $500$ samples from each class. In the 2-class and 5-class cases, the error rates of the SVM classifier are 12.69% and 13.92%, respectively. Our GEM algorithm achieves 0.51% and 2.22%, respectively. These examples show that our method is able to take class predictions produced by a classifier lacking any spatial prior and produce segmentations with a high degree of spatial coherence.

## 6 Conclusions

We have introduced an approach to probabilistic semi-supervised clustering which is particularly suited for image segmentation. The formulation allows supervised, unsupervised, semi-supervised, and discriminative modes, and can be used with classical standard image priors (such as Gaussian fields, or wavelet-based priors). Unlike the usual Markov random field approaches, which involve combinatorial optimization, our segmentation algorithm consists of a simple generalized EM algorithm. Several experimental examples illustrated the promising behavior of our method. Ongoing work includes a thorough experimental comparison with state-of-the-art segmentation algorithms, namely, spectral methods [16] and techniques based on "graph-cuts" [19].

**Acknowledgement:** This work was partially supported by the (Portuguese) *Fundação para a Ciência e Tecnologia* (FCT), grant POSC/EEA-SRI/61924/2004.

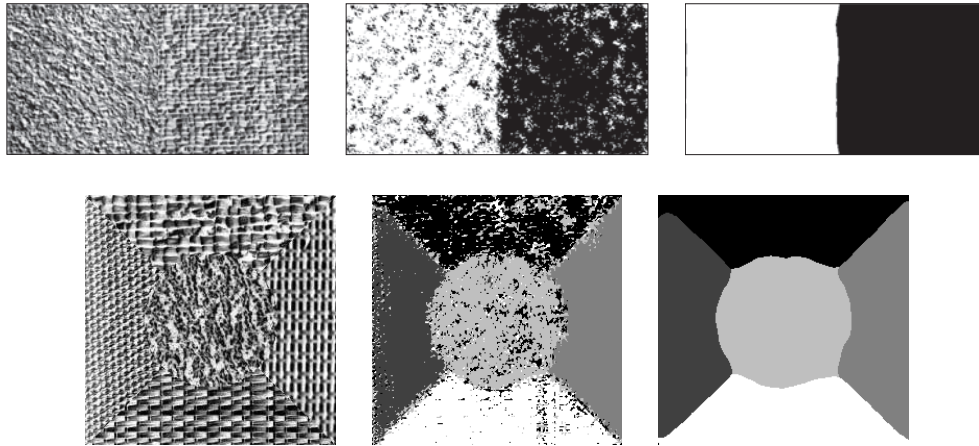

Figure 3: From left to right (in each row): observed image, direct SVM segmentation, segmentation produced by our algorithm.

# References

[1] M. Figueiredo. "Bayesian image segmentation using wavelet-based priors", *Proc. IEEE Conf. Computer Vision and Pattern Recognition - CVPR'2005*, San Diego, CA, 2005.

[2] N. Balram, J. Moura. "Noncausal Gauss-Markov random fields: parameter structure and estimation", *IEEE Trans. Information Theory*, vol. 39, pp. 1333–1355, 1993.

[3] A. Banerjee, S. Merugu, I. Dhillon, J. Ghosh. "Clustering with Bregman divergences." *Proc. SIAM Intern. Conf. Data Mining – SDM'2004*, Lake Buena Vista, FL, 2004.

[4] S. Basu, M. Bilenko, R. Mooney. "A probabilistic framework for semi-supervised clustering." *Proc. of the KDD-2004*, Seattle, WA, 2004.

[5] D. Böhning. "Multinomial logistic regression", *Annals Inst. Stat. Math.*, vol. 44, pp. 197-200, 1992.

[6] G. Chang, B. Yu, M. Vetterli. "Adaptive wavelet thresholding for image denoising and compression." *IEEE Trans. Image Proc.*, vol. 9, pp. 1532–1546, 2000.

[7] T. Hastie, R. Tibshirani, J. Friedman. *The Elements of Statistical Learning*, Springer, 2001.

[8] K. I. Kim, K. Jung, S. H. Park, H. J. Kim. "Support vector machines for texture classification." *IEEE Trans. Pattern Analysis and Machine Intelligence,* vol. 24, pp. 1542–1550, 2002.

[9] D. Hunter, K. Lange. "A tutorial on MM algorithms", *The American Statistician*, vol. 58, pp. 30–37, 2004.

[10] M. Law, A. Topchy, A. K. Jain. "Model-based clustering with probabilistic constraints." In *Proc. of the SIAM Conf. on Data Mining*, pp. 641-645, Newport Beach, CA, 2005.

[11] Z. Lu, T. Leen. "Probabilistic penalized clustering." In *NIPS 17*, MIT Press, 2005.

[12] S. Mallat. *A Wavelet Tour of Signal Proc.*. Academic Press, San Diego, CA, 1998.

[13] G. McLachlan, T. Krishnan. *The EM Algorithm and Extensions*, Wiley, N. York, 1997.

[14] P. Moulin, J. Liu. "Analysis of multiresolution image denoising schemes using generalized - Gaussian and complexity priors," *IEEE Trans. Inform. Theory,* vol. 45, pp. 909–919, 1999.

[15] N. Shental, A. Bar-Hillel, T. Hertz, D. Weinshall. "Computing Gaussian mixture models with EM using equivalence constraints." In *NIPS 15*, MIT Press, Cambridge, MA, 2003.

[16] J. Shi, J. Malik, "Normalized cuts and image segmentation." *IEEE-TPAMI*, vol. 22, pp. 888–905, 2000.

[17] K. Wagstaff, C. Cardie, S. Rogers, S. Schrödl. "Constrained K-means clustering with background knowledge." In *Proc. of ICML'2001*, Williamstown, MA, 2001.

[18] C. Wu. "On the convergence properties of the EM algorithm," *Ann. Statistics*, vol. 11, pp. 95-103, 1983.

[19] R. Zabih, V. Kolmogorov, "Spatially coherent clustering with graph cuts." *Proc. IEEE-CVPR*, vol. II, pp. 437–444, 2004.

[20] X. Zhu. "Semi-Supervised Learning Literature Survey", TR-1530, Comp. Sci. Dept., Univ. of Wisconsin, Madison, 2006. Available at www.cs.wisc.edu/~jerryzhu/pub/ssl_survey.pdf
